# Bayesian Transduction

**Thore Graepel, Ralf Herbrich and Klaus Obermayer**
Department of Computer Science
Technical University of Berlin
Franklinstr. 28/29, 10587 Berlin, Germany
{*graepel2, ralfh, oby*} *@cs.tu-berlin.de*

## Abstract

Transduction is an inference principle that takes a training sample and aims at estimating the values of a function at given points contained in the so-called working sample as opposed to the whole of input space for induction. Transduction provides a confidence measure on single predictions rather than classifiers — a feature particularly important for risk-sensitive applications. The possibly infinite number of functions is reduced to a finite number of equivalence classes on the working sample. A rigorous Bayesian analysis reveals that for standard classification loss we cannot benefit from considering more than one test point at a time. The probability of the label of a given test point is determined as the posterior measure of the corresponding subset of hypothesis space. We consider the PAC setting of binary classification by linear discriminant functions (perceptrons) in kernel space such that the probability of labels is determined by the volume ratio in version space. We suggest to sample this region by an ergodic billiard. Experimental results on real world data indicate that Bayesian Transduction compares favourably to the well-known Support Vector Machine, in particular if the posterior probability of labellings is used as a confidence measure to exclude test points of low confidence.

## 1  Introduction

According to Vapnik [9], *when solving a given problem one should avoid solving a more general problem as an intermediate step.* The reasoning behind this principle is that in order to solve the more general task resources may be wasted or compromises may have to be made which would not have been necessary for the solution of the problem at hand. A direct application of this common-sense principle reduces the more general problem of inferring a functional dependency on the whole of input space to the problem of estimating the values of a function at given points (working sample), a paradigm referred to as *transductive inference.* More formally, given a probability measure $\mathbf{P_{XY}}$ on the space of data $\mathcal{X} \times \mathcal{Y} = \mathcal{X} \times \{-1, +1\}$, a *training sample* $S = \{(\mathbf{x}_1, y_1), \dots, (\mathbf{x}_\ell, y_\ell)\}$ is generated i.i.d. according to $\mathbf{P_{XY}}$. Additional $m$ data points $W = \{\mathbf{x}_{\ell+1}, \dots, \mathbf{x}_{\ell+m}\}$ are drawn: the *working sample.* The goal is to label the objects of the working sample $W$ using a fixed set $\mathcal{H}$ of functions

$f : \mathcal{X} \mapsto \{-1, +1\}$ so as to minimise a predefined loss. In contrast, *inductive inference*, aims at choosing a *single* function $f_\ell \in \mathcal{H}$ best suited to capture the dependency expressed by the unknown $\mathbf{P_{XY}}$. Obviously, if we have a transductive algorithm $\mathcal{A}(W, S, \mathcal{H})$ that assigns to each working sample $W$ a set of labels given the training sample $S$ and the set $\mathcal{H}$ of functions, we can define a function $f_S : \mathcal{X} \mapsto \{-1, +1\}$ by $f_S(\mathbf{x}) = \mathcal{A}(\{\mathbf{x}\}, S, \mathcal{H})$ as a result of the transduction algorithm. There are two crucial differences to induction, however: i) $\mathcal{A}(\{\mathbf{x}\}, S, \mathcal{H})$ is not restricted to select a single decision function $f \in \mathcal{H}$ for each $\mathbf{x}$, ii) a transduction algorithm can give performance guarantees on particular labellings instead of functions. In practical applications this difference may be of great importance.

After all, in risk sensitive applications (medical diagnosis, financial and critical control applications) it often matters to know how *confident* we are about a given prediction. In this case a general confidence measure of the classifier w.r.t. the whole input distribution would not provide the desired warranty at all. Note that for linear classifiers some guarantee can be obtained by the margin [7] which in Section 4 we will demonstrate to be too coarse a confidence measure. The idea of transduction was put forward in [8], where also first algorithmic ideas can be found. Later [1] suggested an algorithm for transduction based on linear programming and [3] highlighted the need for confidence measures in transduction.

The paper is structured as follows: A Bayesian approach to transduction is formulated in Section 2. In Section 3 the function class of kernel perceptrons is introduced to which the Bayesian transduction scheme is applied. For the estimation of volumes in parameter space we present a kernel billiard as an efficient sampling technique. Finally, we demonstrate experimentally in Section 4 how the confidence measure for labellings helps Bayesian Transduction to achieve low generalisation error at a low rejection rate of test points and thus to outperform Support Vector Machines (SVMs).

## 2 Bayesian Transductive Classification

Suppose we are given a training sample $S = \{(\mathbf{x}_1, y_1), \ldots, (\mathbf{x}_\ell, y_\ell)\}$ drawn i.i.d. from $\mathbf{P_{XY}}$ and a working sample $W = \{\mathbf{x}_{\ell+1}, \ldots, \mathbf{x}_{\ell+m}\}$ drawn i.i.d. from $\mathbf{P_X}$. Given a *prior* $\mathbf{P_H}$ over the set $\mathcal{H}$ of functions and a likelihood $\mathbf{P}_{(XY)^\ell|H=f}$ we obtain a posterior probability $\mathbf{P}_{H|(XY)^\ell=S} \overset{\text{def}}{=} \mathbf{P}_{H|S}$ by Bayes' rule. This posterior measure induces a probability measure on labellings $\mathbf{b} \in \{-1, +1\}^m$ of the working sample by[1]

$$\mathbf{P}_{Y^m|S,W}(\mathbf{b}) \overset{\text{def}}{=} \mathbf{P}_{H|S}(\{f : \forall \mathbf{x}_{\ell+i} \in W \quad f(\mathbf{x}_{\ell+i}) = b_i\}) . \tag{1}$$

For the sake of simplicity let us assume a PAC style setting, i.e. there exists a function $f^*$ in the space $\mathcal{H}$ such that $\mathbf{P}_{Y|X=\mathbf{x}}(y) = \delta(y - f^*(\mathbf{x}))$. In this case one can define the so-called *version-space* as the set of functions that is consistent with the training sample

$$V(S) = \{f : \forall (\mathbf{x}_i, y_i) \in S \quad f(\mathbf{x}_i) = y_i\} , \tag{2}$$

outside which the posterior $\mathbf{P}_{H|S}$ vanishes. Then $\mathbf{P}_{Y^m|S,W}(\mathbf{b})$ represents the prior measure of functions consistent with the training sample $S$ *and* the labelling $\mathbf{b}$ on the working sample $W$ normalised by the prior measure of functions consistent with $S$ alone. The measure $\mathbf{P_H}$ can be used to incorporate prior knowledge into

the inference process. If no such knowledge is available, considerations of symmetry may lead to "uninformative" priors.

Given the measure $\mathbf{P}_{\mathsf{Y}^m|S,W}$ over labellings, in order to arrive at a risk minimal decision w.r.t. the labelling we need to define a loss function $l : \mathcal{Y}^m \times \mathcal{Y}^m \mapsto \mathbb{R}^+$ between labellings and minimise its expectation,

$$R(\mathbf{b}, S, W) = \mathbf{E}_{\mathsf{Y}^m|S,W}\left[l(\mathbf{b}, \mathsf{Y}^m)\right] = \sum_{\{\mathbf{b}'\}} l(\mathbf{b}, \mathbf{b}') \mathbf{P}_{\mathsf{Y}^m|S,W}(\mathbf{b}') , \qquad (3)$$

where the summation runs over all the $2^m$ possible labellings $\mathbf{b}'$ of the working sample. Let us consider two scenarios:

1. A 0–1–loss on the exact labelling $\mathbf{b}$, i.e. for two labellings $\mathbf{b}$ and $\mathbf{b}'$

$$l_c(\mathbf{b}, \mathbf{b}') = 1 - \prod_{i=1}^m \delta(b_i - b_i') \quad \Leftrightarrow \quad R_c(\mathbf{b}, S, W) = 1 - \mathbf{P}_{\mathsf{Y}^m|S,W}(\mathbf{b}) . \quad (4)$$

   In this case choosing the labelling $\mathbf{b}_c = \mathrm{argmin}_{\mathbf{b}} R_c(\mathbf{b}, S, W)$ of the highest joint probability $\mathbf{P}_{\mathsf{Y}^m|S,W}(\mathbf{b})$ minimises the risk. This non-labelwise loss is appropriate if the goal is to *exactly* identify a combination of labels, e.g. the combination of handwritten digits defining a postal zip code. Note that classical SVM transduction (see, e.g. [8, 1]) by maximising the margin on the combined training and working sample approximates this strategy and hence does not minimise the standard classification risk on single instances as intended.

2. A 0–1–loss on the single labels $b_i$, i.e. for two labellings $\mathbf{b}$ and $\mathbf{b}'$

$$l_s(\mathbf{b}, \mathbf{b}') = \frac{1}{m} \sum_{i=1}^m (1 - \delta(b_i - b_i')) , \qquad (5)$$

$$R_s(\mathbf{b}, S, W) = \frac{1}{m} \sum_{i=1}^m \sum_{\{\mathbf{b}'\}} (1 - \delta(b_i - b_i')) \mathbf{P}_{\mathsf{Y}^m|S,W}(\mathbf{b}')$$

$$= \frac{1}{m} \sum_{i=1}^m \left(1 - \mathbf{P}_{\mathsf{H}|S}(\{f : f(\mathbf{x}_{\ell+i}) = b_i\})\right) .$$

   Due to the independent treatment of the loss at working sample points the risk $R_s(\mathbf{b}, S, W)$ is minimised by the labelling of highest marginal probability of the labels, i.e.

$$b_i = \mathrm{argmax}_{y \in \mathcal{Y}} \mathbf{P}_{\mathsf{H}|S}(\{f : f(\mathbf{x}_{\ell+i}) = y\}) .$$

   Thus in the case of the labelwise loss (5) a working sample of $m > 1$ point does not offer any advantages over larger working samples w.r.t. the Bayes-optimal decision. Since this corresponds to the standard classification setting, we will restrict ourselves to working samples of size $m = 1$, i.e. to one working point $\mathbf{x}_{\ell+1}$.

## 3    Bayesian Transduction by Volume

### 3.1    The Kernel Perceptron

We consider transductive inference for the class of kernel perceptrons. The decision functions are given by

$$f(\mathbf{x}) = \mathrm{sign}\left(\langle \mathbf{w}, \phi(\mathbf{x})\rangle_{\mathcal{F}}\right) = \mathrm{sign}\left(\sum_{i=1}^\ell \alpha_i k(\mathbf{x}_i, \mathbf{x})\right) \qquad \mathbf{w} = \sum_{i=1}^\ell \alpha_i \phi(\mathbf{x}_i) \in \mathcal{F} ,$$

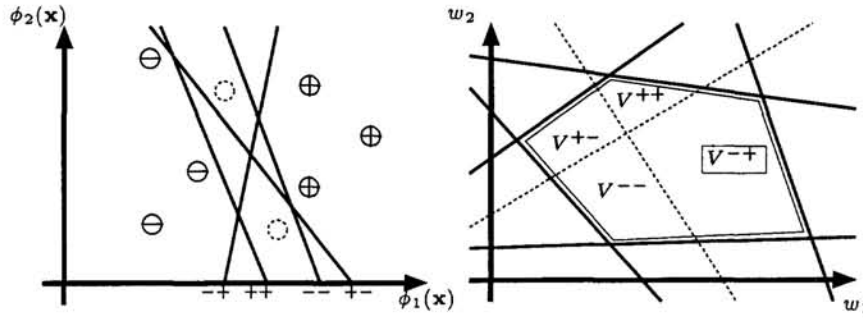

Figure 1: Schematic view of data space (left) and parameter space (right) for a classification toy example. Using the duality given by $\langle \mathbf{w}, \phi(\mathbf{x}) \rangle_{\mathcal{F}} = 0$ data points on the left correspond to hyperplanes on the right, while hyperplanes on the left can be thought of as points on the right.

where the mapping $\phi : \mathcal{X} \mapsto \mathcal{F}$ maps from input space $\mathcal{X}$ to a *feature space* $\mathcal{F}$ completely determined by the inner product function (*kernel*) $k : \mathcal{X} \times \mathcal{X} \mapsto \mathbb{R}$ (see [9, 10]). Given a training sample $S = \{(\mathbf{x}_i, y_i)\}_{i=1}^{\ell}$ we can define the version space — the set of all perceptrons compatible with the training data — as in (2) having the additional constraint $\|\mathbf{w}\|_{\mathcal{F}} = 1$ ensuring uniqueness. In order to obtain a prediction on the label $b_1$ of the working point $\mathbf{x}_{\ell+1}$ we note that $\mathbf{x}_{\ell+1}$ may bisects the volume $V$ of version space into two sub–volumes $V^+$ and $V^-$, where the perceptrons in $V^+$ would classify $\mathbf{x}_{\ell+1}$ as $b_1 = +1$ and those in $V^-$ as $b_1 = -1$. The ratio $p^+ = V^+/V$ is the probability of the labelling $b_1 = +1$ given a uniform prior $\mathbf{P}_H$ over $\mathbf{w}$ and the class of kernel perceptrons, accordingly for $b_1 = -1$ (see Figure 1). Already Vapnik in [8, p. 323] noticed that it is troublesome to estimate sub–volumes of version space. As the solution to this problem we suggest to use a billiard algorithm.

## 3.2 Kernel Billiard for Volume Estimation

The method of playing billiard in version space was first introduced by Rujan [6] for the purpose of estimating its centre of mass and consequently refined and extended to kernel spaces by [4]. For Bayesian Transduction the idea is to bounce the billiard ball in version space and to record how much time it spends in each of the sub-volumes of interest. Under the assumption of ergodicity [2] w.r.t. the uniform measure in the limit the accumulated flight times for each sub-volume are proportional to the sub-volume itself.

Since the trajectory is located in $\mathcal{F}$ each position $\mathbf{w}$ and direction $\mathbf{v}$ of the ball can be expressed as linear combinations of the $\phi(\mathbf{x}_i)$, i.e.

$$\mathbf{w} = \sum_{i=1}^{\ell} \alpha_i \phi(\mathbf{x}_i) \quad \mathbf{v} = \sum_{i=1}^{\ell} \beta_i \phi(\mathbf{x}_i) \quad \langle \mathbf{w}, \mathbf{v} \rangle_{\mathcal{F}} = \sum_{i,j=1}^{\ell} \alpha_i \beta_j k(\mathbf{x}_i, \mathbf{x}_j)$$

where $\alpha, \beta$ are real vectors with $\ell$ components and fully determine the state of the billiard. The algorithm for the determination of the label $b_1$ of $\mathbf{x}_{\ell+1}$ proceeds as follows:

1. Initialise the starting position $\mathbf{w}_0$ in $V(S)$ using any kernel perceptron algorithm that achieves zero training error (e.g. SVM [9]). Set $V^+ = V^- = 0$.

2. Find the closest boundary of $V(S)$ starting from current $\mathbf{w}$ into direction $\mathbf{v}$, where the flight times $\tau_j$ for all points including $\mathbf{x}_{\ell+1}$ are determined using

$$\tau_j = -\frac{\langle \mathbf{w}, \phi(\mathbf{x}_j) \rangle_{\mathcal{F}}}{\langle \mathbf{v}, \phi(\mathbf{x}_j) \rangle_{\mathcal{F}}} .$$

The smallest positive flight time $\tau_c = \min_{j:\tau_j>0} \tau_j$ in kernel space corresponds to the closest data point boundary $\phi(\mathbf{x}_c)$ on the hypersphere. Note, that if $\tau_c \to \infty$ we randomly generate a direction $\mathbf{v}$ pointing *towards* version space, i.e. $y \langle \mathbf{v}, \phi(\mathbf{x}) \rangle_{\mathcal{F}} > 0$ assuming the last bounce was at $\phi(\mathbf{x})$.

3. Calculate the ball's new position $\mathbf{w}'$ according to

$$\mathbf{w}' = \frac{\mathbf{w} + \tau_c \mathbf{v}}{\|\mathbf{w} + \tau_c \mathbf{v}\|_{\mathcal{F}}} .$$

Calculate the distance $t_i^y = \|\mathbf{w} - \mathbf{w}'\|_{\text{sphere}} = \arccos \left( 1 - \|\mathbf{w} - \mathbf{w}'\|_{\mathcal{F}}^2 / 2 \right)$ on the hypersphere and add it to the volume estimate $V^y$ corresponding to the current label $y = \text{sign} \left( \langle \mathbf{w} + \mathbf{w}', \phi(\mathbf{x}_{\ell+1}) \rangle_{\mathcal{F}} \right)$. If the test point $\phi(\mathbf{x}_{\ell+1})$ was hit, i.e. $c = \ell + 1$, keep the old direction vector $\mathbf{v}$. Otherwise update to the reflection direction $\mathbf{v}'$,

$$\mathbf{v}' = \mathbf{v} - 2 \langle \mathbf{v}, \phi(\mathbf{x}_c) \rangle_{\mathcal{F}} \phi(\mathbf{x}_c) .$$

Go back to step 2 unless the stopping criterion (8) is met.

Note that in practice one trajectory can be calculated in advance and can be used for all test points. The estimators of the probability of the labellings are then given by $\widehat{p}^+ = V^+/(V^+ + V^-)$ and $\widehat{p}^- = V^-/(V^+ + V^-)$. Thus, the algorithm outputs $\widehat{b}_1$ with confidence $\widehat{c}_{\text{trans}}$ according to

$$\widehat{b}_1 \overset{\text{def}}{=} \text{argmax}_{y \in \mathcal{Y}} \, \widehat{p}^y , \tag{6}$$

$$\widehat{c}_{\text{trans}} \overset{\text{def}}{=} \left( 2 \cdot \max\left(\widehat{p}^+, \widehat{p}^-\right) - 1 \right) \in [0,1] . \tag{7}$$

Note that the Bayes Point Machine (BPM) [4] aims at an optimal approximation of the transductive classification (6) by a single function $f \in \mathcal{H}$ and that the well known SVM can be viewed as an approximation of the BPM by the centre of the largest ball in version space. Thus, treating the real valued output $|f(\mathbf{x}_{\ell+1})| \overset{\text{def}}{=} \widehat{c}_{\text{ind}}$ of SVM classifiers as a confidence measure can be considered an approximation of (7). The consequences will be demonstrated experimentally in the following section.

Disregarding the issue of mixing time [2] and the dependence of trajectories we assume for the stopping criterion that the fraction $p_i^+$ of time $t_i^+$ spent in volume $V^+$ on trajectory $i$ of length $\left(t_i^+ + t_i^-\right)$ is a random variable having expectation $p^+$. Hoeffding's inequality [5] bounds the probability of deviation from the expectation $p^+$ by more than $\epsilon$,

$$\mathbf{P}\left( \frac{1}{n} \sum_{i=1}^{n} p_i^+ - p^+ \geq \epsilon \right) \leq \exp\left(-2n\epsilon^2\right) \overset{\text{def}}{=} \eta . \tag{8}$$

Thus if we want the deviation $\epsilon$ from the true label probability to be less than $\epsilon < 0.05$ with probability at least $1 - \eta = 0.99$ we need approximately $n \approx 1000$ bounces. The computational effort of the above algorithm for a working set of size $m$ is of order $\mathcal{O}(n\ell(m + \ell))$.

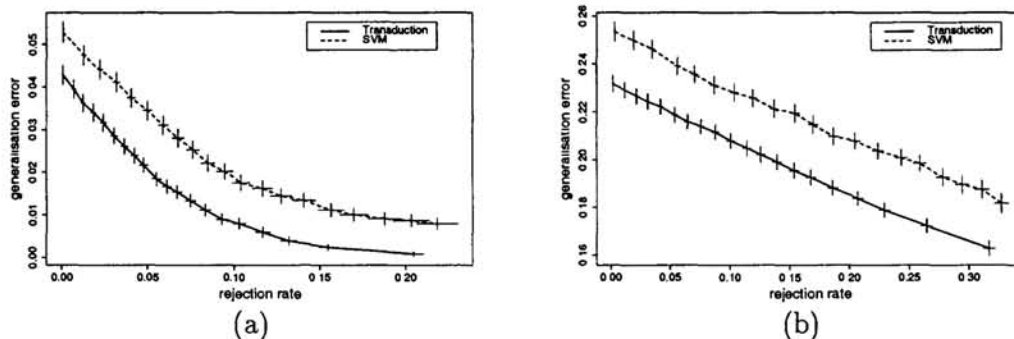

Figure 2: Generalisation error vs. rejection rate for Bayesian Transduction and SVMs for the **thyroid** data set ($\sigma = 3$) **(a)** and the **heart** data set ($\sigma = 10$). The error bars in both directions indicate one standard deviation of the estimated means. The upper curve depicts the result for the SVM algorithm; the lower curve is the result obtained by Bayesian Transduction.

## 4   Experimental Results

We focused on the confidence $\widehat{c}_{\text{trans}}$ Bayesian Transduction provides together with the prediction $\widehat{b}_1$ of the label. If the confidence $\widehat{c}_{\text{trans}}$ reflects reliability of a label estimate at a given test point then rejecting those test points whose predictions carry low confidence should lead to a reduction in generalisation error on the remaining test points. In the experiments we varied a rejection threshold $\theta$ between $[0, 1]$ thus obtaining for each $\theta$ a rejeection rate together with an estimate of the generalisation error at non-rejected points. Both these curves were linked by their common $\theta$-axis resulting in a generalisation error versus rejection rate plot.

We used the UCI[2] data sets **thyroid** and **heart** because they are medical applications for which the confidence of single predictions is particularly important. Also a high rejection rate due to too conservative a confidence measure may incur considerable costs. We trained a Support Vector Machine using RBF kernels $k\left(\mathbf{x}, \mathbf{x}'\right) = \exp\left(-\|\mathbf{x} - \mathbf{x}'\|^2 / 2\sigma^2\right)$ with $\sigma$ chosen such as to insure the existence of a version space. We used 100 different training samples obtained by random 60%:40% splits of the whole data set. The margin $\widehat{c}_{\text{ind}}$ of each test point was calculated as a confidence measure of SVM classifications. For comparison we determined the labels $\widehat{b}_1$ and resulting confidences $\widehat{c}_{\text{trans}}$ using the Bayesian Transduction algorithm (see Section 3) with the same value of the kernel parameter. Since the rejection for the Bayesian Transduction was in both cases higher than for SVMs at the same level $\theta$ we determined $\theta_{\max}$ which achieves the same rejection rate for the SVM confidence measures as Bayesian Transduction achieves at $\theta = 1$ (**thyroid**: $\theta_{\max} = 2.15$, **heart**: $\theta_{\max} = 1.54$). The results for the two data sets are depicted in Figure 2.

In the **thyroid** example Figure 2 (a) one can see that $\widehat{c}_{\text{trans}}$ is indeed an appropriate indicator of confidence: at a rejection rate of approximately 20% the generalisation error approaches zero at minimal variance. For any desired generalisation error Bayesian Transduction needs to reject significantly less examples of the test set as compared to SVM classifiers, e.g. 4% less at 2.3% generalisation error. The results of the **heart** data set show even more pronounced characteristics w.r.t. to the rejection

rate. Note that those confidence measures considered cannot capture the effects of noise in the data which leads to a generalisation error of 16.4% even at maximal rejection $\theta = 1$ corresponding to the Bayes error under the given function class.

## 5 Conclusions and Future Work

In this paper we a presented a Bayesian analysis of transduction. The required volume estimates for kernel perceptrons in version space are performed by an ergodic billiard in kernel space. Most importantly, transduction not only determines the label of a given point but also returns a confidence measure of the classification in the form of the probability of the label under the model. Using this confidence measure to reject test examples then lead to improved generalisation error over SVMs. The billiard algorithm can be extended to the case of non-zero training error by allowing the ball to penetrate walls, a property that is captured by adding a constant $\lambda$ to the diagonal of the kernel matrix [4]. Further research will aim at the discovery of PAC-Bayesian bounds on the generalisation error of transduction.

## Acknowledgements

We are greatly indebted to U. Kockelkorn for many interesting suggestions and discussions. This project was partially funded by Technical University of Berlin via FIP 13/41.

## Footnotes

[1]Note that the number of different labellings $\mathbf{b}$ implementable by $\mathcal{H}$ is bounded above by the value of the growth function $\Pi_\mathcal{H}(|W|)$ [8, p. 321].

[2]UCI University of California at Irvine: Machine Learning Repository

## References

[1] K. Bennett. *Advances in Kernel Methods — Support Vector Learning*, chapter 19, Combining Support Vector and Mathematical Programming Methods for Classification, pages 307–326. MIT Press, 1998.

[2] I. Cornfeld, S. Fomin, and Y. Sinai. *Ergodic Theory*. Springer Verlag, 1982.

[3] A. Gammerman, V. Vovk, and V. Vapnik. Learning by transduction. In *Proceedings of Uncertainty in AI*, pages 148–155, Madison, Wisconsin, 1998.

[4] R. Herbrich, T. Graepel, and C. Campbell. Bayesian learning in reproducing kernel Hilbert spaces. Technical report, Technical University Berlin, 1999. TR 99-11.

[5] W. Hoeffding. Probability inequalities for sums of bounded random variables. *Journal of the American Statistical Association*, 58:13–30, 1963.

[6] P. Ruján. Playing billiard in version space. *Neural Computation*, 9:99–122, 1997.

[7] J. Shawe-Taylor. Confidence estimates of classification accuracy on new examples. Technical report, Royal Holloway, University of London, 1996. NC2–TR–1996–054.

[8] V. Vapnik. *Estimation of Dependences Based on Empirical Data*. Springer, 1982.

[9] V. Vapnik. *The Nature of Statistical Learning Theory*. Springer, 1995.

[10] G. Wahba. *Spline Models for Observational Data*. Society for Industrial and Applied Mathematics, Philadelphia, 1990.